# A Local Algorithm to Learn Trajectories with Stochastic Neural Networks

Javier R. Movellan*
Department of Cognitive Science
University of California San Diego
La Jolla, CA 92093-0515

## Abstract

This paper presents a simple algorithm to learn trajectories with a continuous time, continuous activation version of the Boltzmann machine. The algorithm takes advantage of intrinsic Brownian noise in the network to easily compute gradients using entirely local computations. The algorithm may be ideal for parallel hardware implementations.

This paper presents a learning algorithm to train continuous stochastic networks to respond with desired trajectories in the output units to environmental input trajectories. This is a task, with potential applications to a variety of problems such as stochastic modeling of neural processes, artificial motor control, and continuous speech recognition. For example, in a continuous speech recognition problem, the input trajectory may be a sequence of fast Fourier transform coefficients, and the output a likely trajectory of phonemic patterns corresponding to the input. This paper was based on recent work on diffusion networks by Movellan and McClelland (in press) and by recent papers by Apolloni and de Falco (1991) and Neal (1992) on asymmetric Boltzmann machines. The learning algorithm can be seen as a generalization of their work to the stochastic diffusion case and to the problem of learning continuous stochastic trajectories.

Diffusion networks are governed by the standard connectionist differential equations plus an independent additive noise component. The resulting process is governed

by a set of Langevin stochastic differential equations

$$da_i(t) = \lambda_i \, drift_i(t) \, dt + \sigma dB_i(t) \; ; \;\; i \in \{1, ..., n\} \tag{1}$$

where $\lambda_i$ is the processing rate of the $i^{th}$ unit, $\sigma$ is the diffusion constant, which controls the flow of entropy throughout the network, and $dB_i(t)$ is a Brownian motion differential (Soon, 1973). The drift function is the deterministic part of the process. For consistency I use the same drift function as in Movellan and McClelland, 1992 but many other options are possible: $drift_i(t) = \sum_{j=1}^{n} w_{ij} a_j(t) - f^{-1} a_i(t)$, where $w_{ij}$ is the weight from the $j^{th}$ to the $i^{th}$ unit, and $f^{-1}$ is the inverse of a logistic function scaled in the $(min - max)$ interval: $f^{-1}(a) = log\frac{a-min}{max-a}$.

In practice DNs are simulated in digital computers with a system of stochastic difference equations

$$a_i(t + \Delta t) = a_i(t) + \lambda_i \, drift_i(t) \, \Delta t + \sigma \, z_i(t) \, \sqrt{\Delta t} \; ; \;\; i \in \{1, ..., n\} \tag{2}$$

where $z_i(t)$ is a standard Gaussian random variable. I start the derivations of the learning algorithm for the trajectory learning task using the discrete time process (equation 2) and then I take limits to obtain the continuous diffusion expression. To simplify the derivations I adopt the following notation: a trajectory of states -input, hidden and output units- is represented as $\mathbf{a} = [a(1)...a(t_m)] = [a_1(1)...a_n(1)...a_1(t_m)...a_n(t_m)]$. The trajectory vector can be partitioned into 3 consecutive row vectors representing the trajectories of the input, hidden and output units $\mathbf{a} = [\mathbf{xhy}]$.

The key to the learning algorithm is obtaining the gradient of the probability of specific trajectories. Once we know this gradient we have all the information needed to increase the probability of desired trajectories and decrease the probability of unwanted trajectories. To obtain this gradient we first need to do some derivations on the transition probability densities. Using the discrete time approximation to the diffusion process, it follows that the conditional transition probability density functions are multivariate Gaussian

$$p(\mathbf{a}(t + \Delta t)|\mathbf{a}(t)) = \prod_{i=1}^{n} \frac{1}{\sqrt{2\pi \Delta t}\sigma} e^{\frac{-z_i^2(t)}{2\sigma\sqrt{\Delta t}}} \tag{3}$$

From equation 2 and 3 it follows that

$$\frac{\partial}{\partial w_{ij}} log \, p(\mathbf{a}(t + \Delta t)|\, \mathbf{a}(t)) = \frac{\lambda_i}{\sigma} z_i(t) \, \sqrt{\Delta t} a_j(t) \tag{4}$$

Since the network is Markovian, the probability of an entire trajectory can be computed from the product of the transition probabilities

$$p(\mathbf{a}) = p(\mathbf{a}(t_0)) \prod_{t=t_0}^{t_{m-1}} p(\mathbf{a}(t + \Delta t)|\mathbf{a}(t)) \tag{5}$$

The derivative of the probability of a specific trajectory follows

$$\frac{\partial p(\mathbf{a})}{\partial w_{ij}} = p(\mathbf{a})\frac{\lambda_i}{\sigma} \sum_{t=t_0}^{t_{m-1}} z_i(t) \, \sqrt{\Delta t} \, a_j(t) \tag{6}$$

In practice, the above rule is all is needed for discrete time computer simulations. We can obtain the continuous time form by taking limits as $\Delta t \to 0$, in which case the sum becomes Ito's stochastic integral of $a_j(t)$ with respect to the Brownian motion differential over a $\{t_o, T\}$ interval.

$$\frac{\partial p(\mathbf{a})}{\partial w_{ij}} = p(\mathbf{a})\frac{\lambda_i}{\sigma} \int_{t_0}^{T} a_j(t)dB_i(t) \tag{7}$$

A similar equation may be obtained for the $\lambda_i$ parameters

$$\frac{\partial p(\mathbf{a})}{\partial \lambda_i} = p(\mathbf{a})\frac{1}{\sigma} \int_{t_0}^{T} drift_i(t)dB_i(t) \tag{8}$$

For notational convenience I define the following random variables and refer to them as the *delta signals*

$$\delta_{w_{ij}}(\mathbf{a}) = \frac{\partial log\ p(\mathbf{a})}{\partial w_{ij}} = \frac{\lambda_i}{\sigma} \int_{t_0}^{T} a_j(t)dB_i(t) \tag{9}$$

and

$$\delta_{\lambda_i}(\mathbf{a}) = \frac{\partial log\ p(\mathbf{a})}{\partial \lambda_i} = \frac{1}{\sigma} \int_{t_0}^{T} drift_i(t)dB_i(t) \tag{10}$$

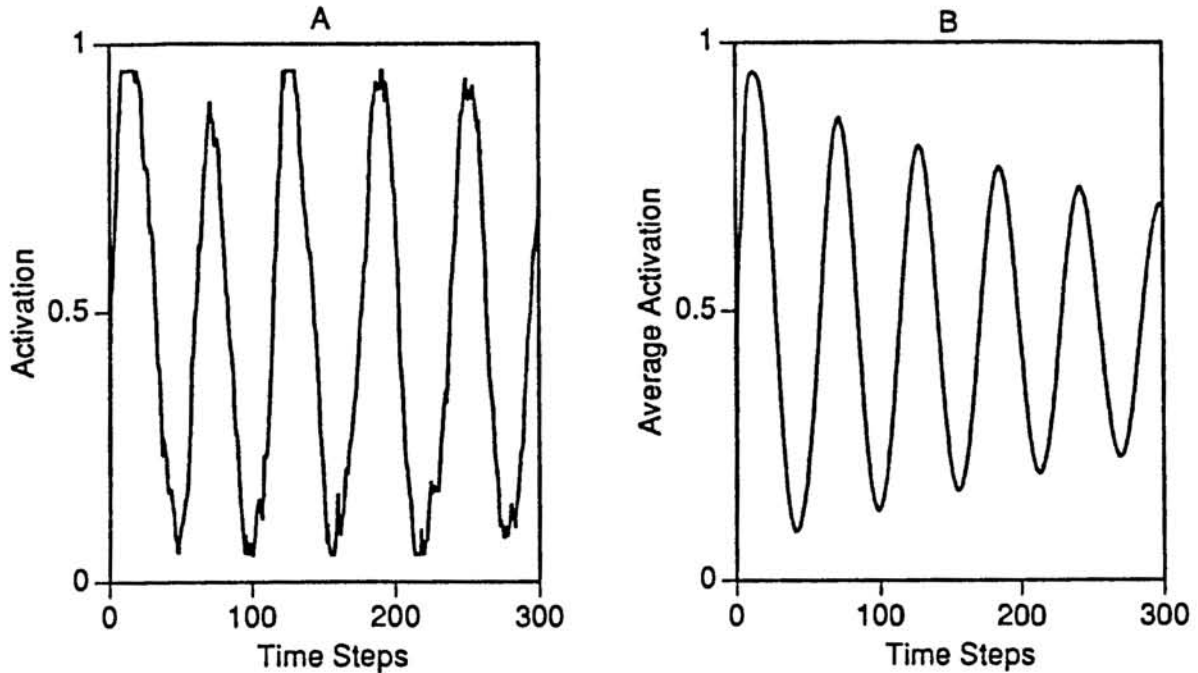

Figure 1: A) A sample Trajectory. B) The Average Trajectory. As Time Progresses Sample Trajectories Become Statistically Independent Dampening the Average.

The approach taken in this paper is to minimize the expected value of the error assigned to spontaneously generated trajectories $O = E(\rho(\mathbf{a}))$ where $\rho(\mathbf{a})$ is a signal indicating the overall error of a particular trajectory and usually depends only on the output unit trajectory. The necessary gradients follow

$$\frac{\partial O}{\partial w_{ij}} = E(\delta_{w_{ij}}\rho) \qquad (11)$$

$$\frac{\partial O}{\partial \lambda_i} = E(\delta_{\lambda_i}\rho) \qquad (12)$$

Since the above learning rule does not require calculating derivatives of the $\rho$ function, it provides great flexibility making it applicable to a wide variety of situations. For example $\rho(\mathbf{a})$ can be the TSS between the desired and obtained output unit trajectories or it could be a reinforcement signal indicating whether the trajectory is or is not desirable. Figure 1.a shows a typical output of a network trained with TSS as the $\rho$ signal to follow a sinusoidal trajectory. The network consisted of 1 input unit, 3 hidden units, and 1 output unit. The input was constant through time and the network was trained only with the first period of the sinusoid. The expected values in equations 11 and 12 were estimated using 400 spontaneously generated trajectories at each learning epoch. It is interesting to note that although the network was trained for a single period, it continued oscillating without dampening. However, the expected value of the activations dampened, as Figure 1.b shows. The dampening of the average activation is due to the fact that as time progresses, the effects of noise accumulate and the initially phase locked trajectories become independent oscillators.

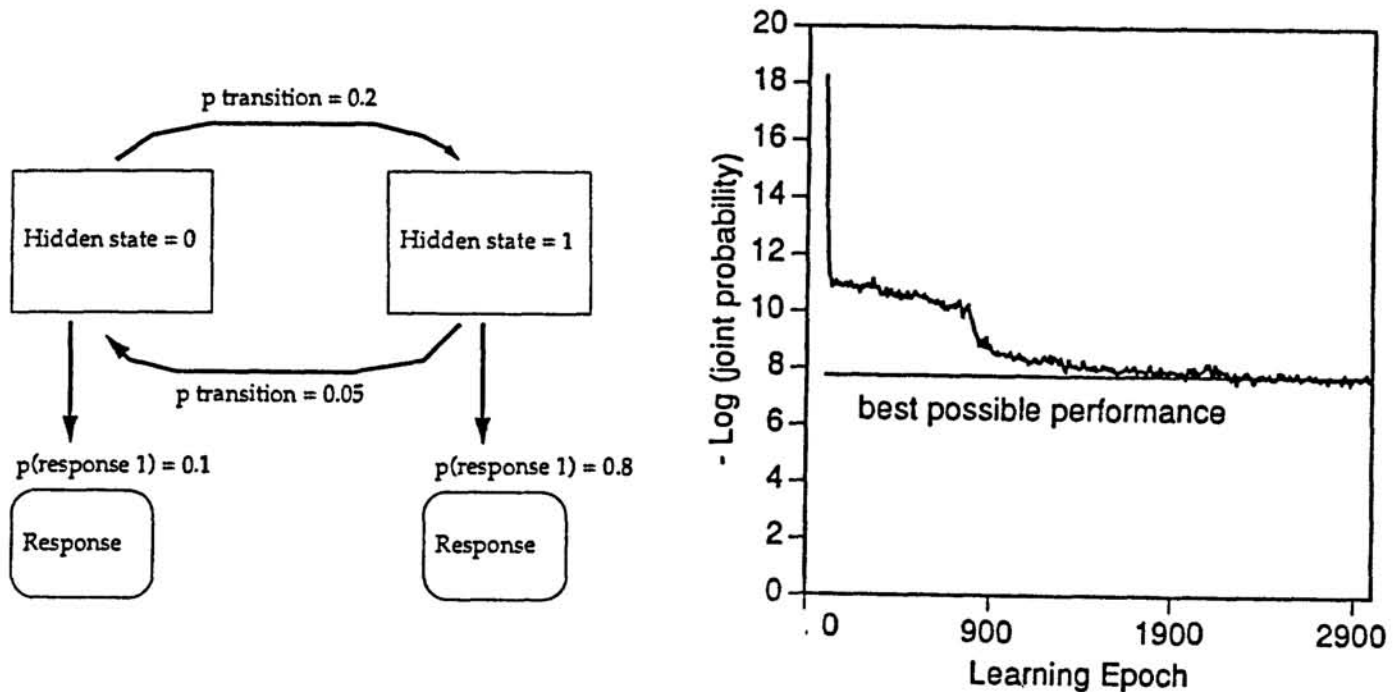

Figure 2: A) The Hidden Markov Emitter. B) Average Error Throughout Training. The Bayesian Limit is Achieved at About 2000 Epochs.

The learning rule is also applicable in reinforcement situations where we just have an overall measure of fitness of the obtained trajectories, but we do not know what the desired trajectory looks like. For example, in a motor control problem we could use as fitness signal $(-\rho)$ the distance walked by a robot controlled by a DN network. Equations 11 and 12 could then be used to gradually improve the average distance walked by the robot. In trajectory recognition problems we could use an overall judgment of the likelihood of the obtained trajectories. I tried this last approach with a toy version of a continuous speech recognition problem. The "emitter" was a hidden Markov model (see Figure 2) that produced sequences of outputs - the equivalent of fast Fourier transform loads - fed as input to the receiver. The receiver was a DN network which received as input, sequences of 10 outputs from the emitter Markov model. The network's task was to guess the sequence of hidden states of the emitter given the sequence of outputs from the emitter. The DN outputs were interpreted as the inferred state of the emitter. Output unit activations greater than 0.5 were evaluated as indicating that the emitter was in state 1 at that particular time. Outputs smaller than 0.5 were evaluated as state 0. To achieve optimal performance in this task the network had to combine two sources of information: top-down information about typical state transitions of the emitter, and bottom up information about the likelihood of the hidden states of the emitter given its responses.

The network was trained with rules 11 and 12 using the negative log joint probability of the DN input trajectory and the DN output trajectory as error signal. This signal was calculated using the transition probabilities of the emitter hidden Markov model and did not require knowledge of its actual state trajectories. The necessary gradients for equations 11 and 12 were estimated using 1000 spontaneous trajectories at each learning epoch. As Figure 3 shows the network started producing unlikely trajectories but continuously improved. The figure also shows the performance expected from an optimal classifier. As training progressed the network approached optimal performance.

## Acknowledgements

This work was funded through the NIMH grant MH47566 and a grant from the Pittsburgh Supercomputer Center.

## Footnotes

*Part of this work was done while at Carnegie Mellon University.

## References

B. Apolloni, & D. de Falco. (1991) Learning by asymmetric parallel Boltzmann machines. *Neural Computation*, **3**, 402-408.

R. Neal. (1992) Asymmetric Parallel Boltzmann Machines are Belief Networks, *Neural Computation*, **4**, 832-834.

J. Movellan & J. McClelland. (1992a) Learning continuous probability distributions with symmetric diffusion networks. To appear in *Cognitive Science*.

T. Soon. (1973) *Random Differential Equations in Science and Engineering*, Academic Press, New York.